# Fast and Accurate $k$-means For Large Datasets

**Michael Shindler**
School of EECS
Oregon State University
shindler@eecs.oregonstate.edu

**Alex Wong**
Department of Computer Science
UC Los Angeles
alexw@seas.ucla.edu

**Adam Meyerson**
Google, Inc.
Mountain View, CA
awmeyerson@google.com

## Abstract

Clustering is a popular problem with many applications. We consider the $k$-means problem in the situation where the data is too large to be stored in main memory and must be accessed sequentially, such as from a disk, and where we must use as little memory as possible. Our algorithm is based on recent theoretical results, with significant improvements to make it practical. Our approach greatly simplifies a recently developed algorithm, both in design and in analysis, and eliminates large constant factors in the approximation guarantee, the memory requirements, and the running time. We then incorporate approximate nearest neighbor search to compute $k$-means in $o(nk)$ (where $n$ is the number of data points; note that computing the cost, given a solution, takes $\Theta(nk)$ time). We show that our algorithm compares favorably to existing algorithms - both theoretically and experimentally, thus providing state-of-the-art performance in both theory and practice.

## 1 Introduction

We design improved algorithms for Euclidean $k$-means in the streaming model. In the $k$-means problem, we are given a set of $n$ points in space. Our goal is to select $k$ points in this space to designate as *facilities* (sometimes called centers or means); the overall cost of the solution is the sum of the squared distances from each point to its nearest facility. The goal is to minimize this cost; unfortunately the problem is **NP**-Hard to optimize, although both heuristic [21] and approximation algorithm techniques [20, 25, 7] exist. In the *streaming model*, we require that the point set be read sequentially, and that our algorithm stores very few points at any given time. Many problems which are easy to solve in the standard batch-processing model require more complex techniques in the streaming model (a survey of streaming results is available [3]); nonetheless there are a number of existing streaming approximations for Euclidean $k$-means. We present a new algorithm for the problem based on [9] with several significant improvements; we are able to prove a faster worst-case running time and a better approximation factor. In addition, we compare our algorithm empirically with the previous state-of-the-art results of [2] and [4] on publicly available large data sets. Our algorithm outperforms them both.

The notion of clustering has widespread applicability, such as in data mining, pattern recognition, compression, and machine learning. The $k$-means objective is one of the most popular formalisms, and in particular Lloyd's algorithm [21] has significant usage [5, 7, 19, 22, 23, 25, 27, 28]. Many of the applications for $k$-means have experienced a large growth in data that has overtaken the amount of memory typically available to a computer. This is expressed in the *streaming model*, where an algorithm must make one (or very few) passes through the data, reflecting cases where random access to the data is unavailable, such as a very large file on a hard disk. Note that the data size, despite being large, is still finite.

Our algorithm is based on the recent work of [9]. They "guess" the cost of the optimum, then run the online facility location algorithm of [24] until either the total cost of the solution exceeds a constant times the guess or the total number of facilities exceeds some computed value $\kappa$. They then declare the end of a *phase*, increase the guess, consolidate the facilities via matching, and continue with the next point. When the stream has been exhausted, the algorithm has some $\kappa$ facilities, which are then consolidated down to $k$. They then run a ball $k$-means step (similar to [25]) by maintaining samples of the points assigned to each facility and moving the facilities to the centers of mass of these samples. The algorithm uses $O(k \log n)$ memory, runs in $O(nk \log n)$ time, and obtains an $O(1)$ worst-case approximation. Provided that the original data set was $\sigma$-separable (see section 1.2 for the definition), they use ball $k$-means to improve the approximation factor to $1 + O(\sigma^2)$.

From a practical standpoint, the main issue with [9] is that the constants hidden in the asymptotic notation are quite large. The approximation factor is in the hundreds, and the $O(k \log n)$ memory requirement has sufficiently high constants that there are actually more than $n$ facilities for many of the data sets analyzed in previous papers. Further, these constants are encoded into the algorithm itself, making it difficult to argue that the performance should improve for non-worst-case inputs.

## 1.1 Our Contributions

We substantially simplify the algorithm of [9]. We improve the manner by which the algorithm determines better facility cost as the stream is processed, removing unnecessary checks and allowing the user to parametrize what remains. We show that our changes result in a *better approximation guarantee* than the previous work. We also develop a variant that computes a solution in $o(nk)$ and show experimentally that both algorithms outperform previous techniques.

We remove the end-of-phase condition based on the total cost, ending phases only when the number of facilities exceeds $\kappa$. While we require $\kappa \in \Omega(k \log n)$, we do not require any particular constants in the expression (in fact we will use $\kappa = k \log n$ in our experiments). We also simplify the transition between phases, observing that it's quite simple to bound the number of phases by $\log OPT$ (where $OPT$ is the optimum $k$-means cost), and that in practice this number of phases is usually quite a bit less than $n$.

We show that despite our modifications, the worst case approximation factor is still constant. Our proof is based on a much tighter bound on the cost incurred per phase, along with a more flexible definition of the "critical phase" by which the algorithm should terminate. Our proofs establish that the algorithm converges for any $\kappa > k$; of course, there are inherent tradeoffs between $\kappa$ and the approximation bound. For appropriately chosen constants our approximation factor will be roughly 17, substantially less than the factor claimed in [9] prior to the ball $k$-means step.

In addition, we apply approximate nearest-neighbor algorithms to compute the facility assignment of each point. The running time of our algorithm is dominated by repeated nearest-neighbor calculations, and an appropriate technique can change our running time from $\Theta(nk \log n)$ to $\Theta(n(\log k + \log \log n))$, an improvement for most values of $k$. Of course, this hurts our accuracy somewhat, but we are able to show that we take only a constant-factor loss in approximation. Note that our final running time is actually faster than the $\Theta(nk)$ time needed to compute the $k$-means cost of a given set of facilities!

In addition to our theoretical improvements, we perform a number of empirical tests using realistic data. This allows us to compare our algorithm to previous [4, 2] streaming $k$-means results.

## 1.2 Previous Work

A simple local search heuristic for the $k$-means problem was proposed in 1957 by Lloyd [21]. The algorithm begins with $k$ arbitrarily chosen points as facilities. At each stage, it allocates the points into clusters (each point assigned to closest facility) and then computes the center of mass for each cluster. These become the new facilities for the next phase, and the process repeats until it is stable. Unfortunately, Lloyd's algorithm has no provable approximation bound, and arbitrarily bad examples exist. Furthermore, the worst-case running time is exponential [29]. Despite these drawbacks, Lloyd's algorithm (frequently known simply as k-means) remains common in practice.

The best polynomial-time approximation for $k$-means is by Kanungo, Mount, Netanyahu, Piatko, Silverman, and Wu [20]. Their algorithm uses local search (similar to the $k$-median algorithm of [8]), and is a $9 + \varepsilon$ approximation. However, Lloyd's observed runtime is superior, and this is a high priority for real applications.

Ostrovsky, Rabani, Schulman and Swamy [25] observed that the value of $k$ is typically selected such that the data is "well-clusterable" rather than being arbitrary. They defined the notion of $\sigma$-separability, where the input to $k$-means is said to be $\sigma$-separable if reducing the number of facilities from $k$ to $k-1$ would increase the cost of the optimum solution by a factor $\frac{1}{\sigma^2}$. They designed an algorithm with approximation ratio $1 + O(\sigma^2)$. Subsequently, Arthur and Vassilvitskii [7] showed that the same procedure produces an $O(\log k)$ approximation for arbitrary instances of $k$-means.

There are two basic approaches to the streaming version of the $k$-means problem. Our approach is based on solving $k$-means as we go (thus at each point in the algorithm, our memory contains a current set of facilities). This type of approach was pioneered in 2000 by Guha, Mishra, Motwani, and O'Callaghan [17]. Their algorithm reads the data in blocks, clustering each using some non-streaming approximation, and then gradually merges these blocks when enough of them arrive. An improved result for $k$-median was given by Charikar, O'Callaghan, and Panigrahy in 2003 [11], producing an $O(1)$ approximation using $O(k \log^2 n)$ space. Their work was based on guessing a lower bound on the optimum $k$-median cost and running $O(\log n)$ parallel versions of the online facility location algorithm of Meyerson [24] with facility cost based on the guessed lower bound. When these parallel calls exceeded the approximation bounds, they would be terminated and the guessed lower bound on the optimum $k$-median cost would increase. The recent paper of Braverman, Meyerson, Ostrovsky, Roytman, Shindler, and Tagiku [9] extended the result of [11] to $k$-means and improved the space bound to $O(k \log n)$ by proving high-probability bounds on the performance of online facility location. This result also added a ball $k$-means step (as in [25]) to substantially improve the approximation factor under the assumption that the original data was $\sigma$-separable.

Another recent result for streaming $k$-means, due to Ailon, Jaiswal, and Monteleoni [4], is based on a divide and conquer approach, similar to the $k$-median algorithm of Guha, Meyerson Mishra, Motwani, and O'Callaghan [16]. It uses the result of Arthur and Vassilvitskii [7] as a subroutine, finding $3k \log k$ centers for each block. Their experiment showed that this algorithm is an improvement over an online variant of Lloyd's algorithm and was comparable to the batch version of Lloyd's.

The other approach to streaming $k$-means is based on coresets: selecting a weighted subset of the original input points such that any $k$-means solution on the subset has roughly the same cost as on the original point set. At any point in the algorithm, the memory should contain a weighted representative sample of the points. This approach was first used in a non-streaming setting for a variety of clustering problems by Bădoiu, Har-Peled, and Indyk [10], and in the streaming setting by Har-Peled and Mazumdar [18]; the time and memory bounds were subsequently improved through a series of papers [14, 13] with the current best theoretical bounds by Chen [12]. A practical implementation of the coreset paradigm is due to Ackermann, Lammersen, Märtens, Raupach, Sohler, and Swierkot [2]. Their approach was shown empirically to be fast and accurate on a variety of benchmarks.

## 2 Algorithm and Theory

Both our algorithm and that of [9] are based on the online facility location algorithm of [24]. For the facility location problem, the number of clusters is not part of the input (as it is for $k$-means), but rather a *facility cost* is given; an algorithm to solve this problem may have as many clusters as it desires in its output, simply by denoting some point as a facility. The solution cost is then the sum of the resulting $k$-means cost ("service cost") and the total paid for facilities.

Our algorithm runs the online facility location algorithm of [24] with a small facility cost until we have more than $\kappa \in \Theta(k \log n)$ facilities. It then increases the facility cost, re-evaluates the current facilities, and continues with the stream. This repeats until the entire stream is read. The details of the algorithm are given as Algorithm 1.

The major differences between our algorithm and that of [9] are as follows. We ignore the overall service cost in determining when to end a phase and raise our facility cost $f$. Further, the number of facilities which must open to end a phase can be any $\kappa \in \Theta(k \log n)$, the constants do not depend directly on the competitive ratio of online facility location (as they did in [9]). Finally, we omit the somewhat complicated end-of-phase analysis of [9], which used matching to guarantee that the number of facilities decreased substantially with each phase and allowed bounding the number of phases by $\frac{n}{k \log n}$. We observe that our number of phases will be bounded by $\log_\beta OPT$; while this is not technically bounded in terms of $n$, in practice this term should be smaller than the linear number of phases implied in previous work.

---

**Algorithm 1** Fast streaming $k$-means (data stream, $k$, $\kappa$, $\beta$)

---

1: Initialize $f = 1/(k(1 + \log n))$ and an empty set $K$
2: **while** some portion of the stream remains unread **do**
3:     **while** $|K| \leq \kappa = \Theta(k \log n)$ and some portion of the stream is unread **do**
4:         Read the next point $x$ from the stream
5:         Measure $\delta = \min_{y \in K} d(x, y)^2$
6:         **if** probability $\delta/f$ event occurs **then**
7:             set $K \leftarrow K \bigcup \{x\}$
8:         **else**
9:             assign $x$ to its closest facility in $K$
10:     **if** stream not exhausted **then**
11:         **while** $|K| > \kappa$ **do**
12:             Set $f \leftarrow \beta f$
13:             Move each $x \in K$ to the center-of-mass of its points
14:             Let $w_x$ be the number of points assigned to $x \in K$
15:             Initialize $\hat{K}$ containing the first facility from $K$
16:             **for** each $x \in K$ **do**
17:                 Measure $\delta = \min_{y \in \hat{K}} d(x, y)^2$
18:                 **if** probability $w_x \delta/f$ event occurs **then**
19:                     set $\hat{K} \leftarrow \hat{K} \bigcup \{x\}$
20:                 **else**
21:                     assign $x$ to its closest facility in $\hat{K}$
22:             Set $K \leftarrow \hat{K}$
23:     **else**
24:         Run batch $k$-means algorithm on weighted points $K$
25:         Perform ball $k$-means (as per [9]) on the resulting set of clusters

---

We will give a theoretical analysis of our modified algorithm to obtain a constant approximation bound. Our constant is substantially smaller than those implicit in [9], with most of the loss occurring in the final non-streaming $k$-means algorithm to consolidate $\kappa$ means down to $k$. The analysis will follow from the theorems stated below; proofs of these theorems are deferred to the appendix.

**Theorem 1.** *Suppose that our algorithm completes the data stream when the facility cost is $f$. Then the overall solution prior to the final re-clustering has expected service cost at most $\frac{f \kappa \beta}{\beta - 1}$, and the probability of being within $1 + \epsilon$ of the expected service cost is at least $1 - \frac{1}{poly(n)}$.*

**Theorem 2.** *With probability at least $1 - \frac{1}{poly(n)}$, the algorithm will either halt with $f \leq \frac{\Theta(C^*)\beta}{\kappa}$, where $C^*$ is the optimum $k$-means cost, or it will halt within one phase of exceeding this value. Furthermore, for large values of $\kappa$ and $\beta$, the hidden constant in $\Theta(C^*)$ approaches 4.*

Note that while the worst-case bound of roughly 4 proven here may not seem particularly strong, unlike the previous work of [9], the worst-case performance is not directly encoded into the algorithm. In practice, we would expect the performance of online facility location to be substantially better than worst-case (in fact, if the ordering of points in the stream is non-adversarial there is a proof to this effect in [24]); in addition the assumption was made that distances add (i.e. triangle inequality is tight) which will not be true in practice (especially of points in low-dimensional space). We also assumed that using more than $k$ facilities does not substantially help the optimum service cost (also unlikely to be true for real data). Combining these, it would be unsurprising if our service cost was actually *better than optimum* at the end of the data stream (of course, we used many more facilities than optimum, so it is not precisely a fair comparison). The following theorem summarizes the worst-case performance of the algorithm; its proof is direct from Theorems 1 and 2.

**Theorem 3.** *The cost of our algorithm's final $\kappa$-mean solution is at most $O(C^*)$, where $C^*$ is the cost of the optimum $k$-means solution, with probability $1 - \frac{1}{poly(n)}$. If $\kappa$ is a large constant times $k \log n$ and $\beta > 2$ is fairly large, then the cost of our algorithm's solution will approach $C^* \frac{4\beta^2}{\beta - 1}$; the extra $\beta$ factor is due to "overshooting" the best facility cost $f$.*

We note that if we run the streaming part of the algorithm $M$ times in parallel, we can take the solution with the smallest final facility cost. This improves the approximation factor to roughly $\frac{4\beta^{1+(1/M)}}{\beta-1}$, which approaches 4 in the limit. Of course, increasing $\kappa$ can substantially increase the memory requirement and increasing $M$ can increase both memory and running time requirements.

When the algorithm terminates, we have a set of $\kappa$ weighted means which we must reduce to $k$ means. A theoretically sound approach involves mapping these means back to randomly selected points from the original set (these can be maintained in a streaming manner) and then approximating $k$-means on $\kappa$ points using a non-streaming algorithm. The overall approximation ratio will be twice the ratio established by our algorithm (we lose a factor of two by mapping back to the original points) plus the approximation ratio for the non-streaming algorithm. If we use the algorithm of [20] along with a large $\kappa$, we will get an approximation factor of twice 4 plus $9+\varepsilon$ for roughly 17. Ball $k$-means can then reduce the approximation factor to $1 + O(\sigma^2)$ if the inputs were $\sigma$-separable (as in [25] and [9]; the hidden constant will be reduced by our more accurate algorithm).

## 3 Approximate Nearest Neighbor

The most time-consuming step in our algorithm is measuring $\delta$ in lines 5 and 17. This requires as many as $\kappa$ distance computations; there are a number of results enabling fast computation of approximate nearest neighbors and applying these results will improve our running time. If we can assume that errors in nearest neighbor computation are independent from one point to the next (and that the expected result is good), our analysis from the previous section applies. Unfortunately, many of the algorithms construct a random data structure to store the facilities, then use this structure to resolve all queries; this type of approach implies that errors are *not* independent from one query to the next. Nonetheless we can obtain a constant approximation for sufficiently large choices of $\beta$.

For our empirical result, we will use a very simple approximate nearest-neighbor algorithm based on random projection. This has reasonable performance in expectation, but is not independent from one step to the next. While the theoretical results from this particular approach are not very strong, it works very well in our experiments. For this implementation, a vector $\varpi$ is created, with each of the $d$ dimensions space being chosen independently and uniformly at random from $[0, 1)$. We store our facilities sorted by their inner product with $\varpi$. When a new point $x$ arrives, instead of taking $O(\kappa)$ to determine its (exact) nearest neighbor, we instead use $O(\log \kappa)$ to find the two facilities that $x \cdot \varpi$ is between. We determine the (exact) closer of these two facilities; this determines the value of $\delta$ in lines 5 and 17 and the "closest" facility in lines 9 and 21.

**Theorem 4.** *If our approximate nearest neighbor computation finds a facility with distance at most $\nu$ times the distance to the closest facility in expectation, then the approximation ratio increase by a constant factor.*

We defer explanation of how we form the stronger theoretical result to the appendix.

## 4 Empirical Evaluation

A comparison of algorithms on real data sets gives a great deal of insight as to their relative performance. Real data is not worst-case, implying that neither the asymptotic performance or running-time bounds claimed in theoretical results are necessarily tight. Of course, empirical evaluation depends heavily on the data sets selected for the experiments.

We selected data sets which have been used previously to demonstrate streaming algorithms. A number of the data sets analyzed in previous work were not particularly large, probably so that batch-processing algorithms would terminate quickly on those inputs. The main motivation for streaming is very large data sets, so we are more interested in sets that might be difficult to fit in a main memory and focused on the largest examples. We looked to [2], and used the two biggest data sets they considered. These were the BigCross dataset[1] and the Census1990 dataset [2]. All the other data sets in [2, 4] were either subsets of these or were well under a half million points.

A necessary input for each of these algorithms is the desired number of clusters. Previous work chose $k$ seemingly arbitrarily; typical values were of the form $\{5, 10, 15, 20, 25\}$. While this input provides a well-defined geometry problem, it fails to capture any information about how $k$-means is

used in practice and need not lead to separable data. Instead, we want to select $k$ such that the best $k$-means solution is much cheaper than the best $(k-1)$-means solution. Since $k$-means is **NP**-Hard, we cannot solve large instances to optimality. For the Census dataset we ran several iterations of the algorithm of [25] for each of many values of $k$. We took the best observed cost for each value of $k$, and found the four values of $k$ minimizing the ratio of $k$-means cost to $(k-1)$-means cost.

This was not possible for the larger BigCross dataset. Instead, we ran a modified version of our algorithm; at the end of a phase, it adjusts the facility cost and restarts the stream. This avoids the problem of compounding the approximation factor at the end of a phase. As with Census, we ran this for consecutive values of $k$ and chose the best ratios of observed values; we chose two, rather than four, so that we could finish our experiments in a reasonable amount of time. Our approach to selecting $k$ is closer to what's done in practice, and is more likely to yield meaningful results.

We do not compare to the algorithm of [9]. First, the memory is not configurable, making it not fit into the common baseline that we will define shortly. Second, the memory requirements and runtime, while asymptotically nice, have large leading constants that cause it to be impractical. In fact, it was an attempt to implement this algorithm that initially motivated the work on this paper.

## 4.1 Implementation Discussion

The divide and conquer ("D&C") algorithm [4] can use its available memory in two possible ways. First, it can use the entire amount to read from the stream, writing the results of computing their $3k \log k$ means to disk; when the stream is exhausted, this file is treated as a stream, until an iteration produces a file that fits entirely into main memory. Alternately, the available memory could be partitioned into layers; the first layer would be filled by reading from the stream, and the weighted facilities produced would be stored in the second. When any layer is full, it can be clustered and the result placed in a higher layer, replacing the use of files and disk. Upon completion of the stream, any remaining points are gathered and clustered to produce $k$ final means. When larger amounts of memory are available, the latter method is preferred. With smaller amounts, however, this isn't always possible, and when it is possible, it can produce worse actual running times than a disk-based approach. As our goal is to judge streaming algorithms under low memory conditions, we used the first approach, which is more fitting to such a constraint.

Each algorithm[3] was programmed in C/C++, compiled with g++, and run under Ubuntu Linux (10.04 LTS) on HP Pavilion p6520f Desktop PC, with an AMD Athlon II X4 635 Processor running at 2.9 GhZ and with 6 GB main memory (although nowhere near the entirety of this was used by any algorithm). For StreamKM++, the authors' implementation [2], also in C, was used instead. With all algorithms, the reported cost is determined by taking the resulting $k$ facilities and computing the $k$-means cost across the entire dataset. The time to compute this cost is not included in the reported running times of the algorithms. Each test case was run 10 times and the average costs and running times were reported.

## 4.2 Experimental Design

Our goal is to compare the algorithms at a common basepoint. Instead of just comparing for the same dataset and cluster count, we further constrained each to use the same amount of memory (in terms of number of points stored in random access). The memory constraints were chosen to reflect the usage of small amounts of memory that are close to the algorithms' designers' specifications, where possible. Ailon *et al* [4] suggest $\sqrt{nk}$ memory for the batch process; this memory availability is marked in the charts by an asterisk. The suggestion from [2] for a coreset of size $200k$ was not run for all algorithms, as the amount of memory necessary for computing a coreset of this size is much larger than the other cases, and our goal is to compare the algorithms at a small memory limit. This does produce a drop in solution quality compared to running the algorithm at their suggested parameters, although their approach remains competitive. Finally, our algorithm suggests memory of $\kappa = k \log n$ or a small constant times the same.

In each case, the memory constraint dictates the parameters; for the divide and conquer algorithm, this is simply the batch size. The coreset size is also dictated by the available memory. Our algorithm is a little more parametrizable; when $M$ memory is available, we allowed $\kappa = M/5$ and each facility to have four samples.

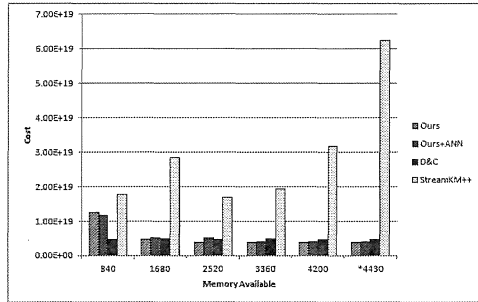

Figure 1: Census Data, k=8, cost

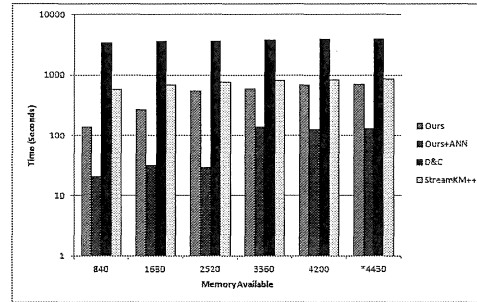

Figure 2: Census Data, k=8, time

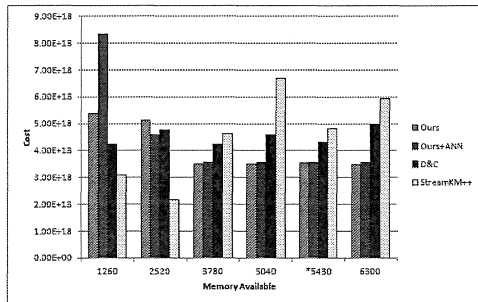

Figure 3: Census Data, k=12, cost

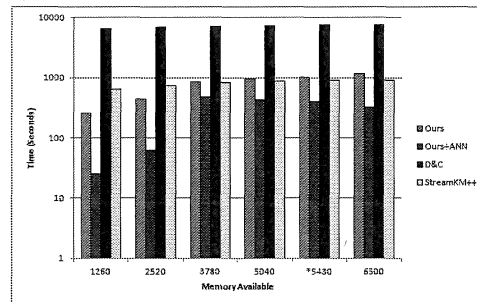

Figure 4: Census Data, k=12, time

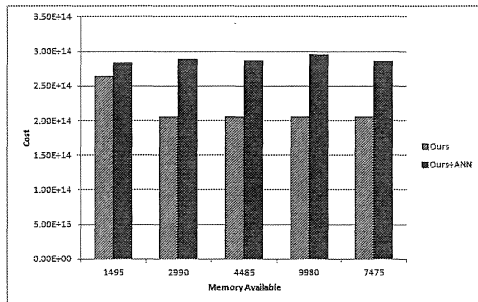

Figure 5: BigCross Data, k=13, cost

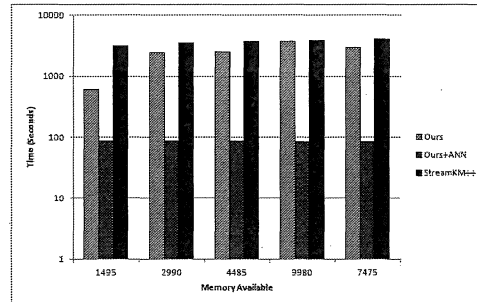

Figure 6: BigCross Data, k=13, time

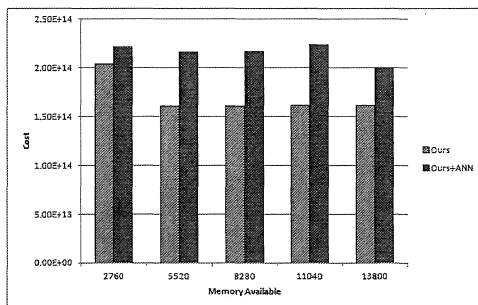

Figure 7: BigCross Data, k=24, cost

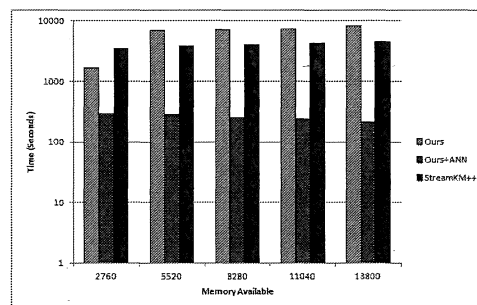

Figure 8: BigCross Data, k=24, time

### 4.3  Discussion of Results

We see that our algorithms are much faster than the D&C algorithm, while having a comparable (and often better) solution quality. We find that we compare well to StreamKM++ in average results, with a closer standard deviation and a better sketch of the data produced. Furthermore, our algorithm stands to gain the most by improved solutions to batch $k$-means, due to the better representative sample present after the stream is processed.

The prohibitively high running time of the divide-and-conquer algorithm [4] is due to the many repeated instances of running their $k$-means# algorithm on each batch of the given size. For sufficiently large memory, this is not problematic, as very few batches will need this treatment. Unfortunately, with very small locally available memory, there will be an immense amount of repeated calls, and the overall running time will suffer greatly. In particular, the observed running time was much worse than the other approaches. For the Census dataset, $k = 12$ case, for example, the slowest run of our algorithm (20 minutes) and the fastest run of the D&C algorithm (125 minutes) occurred at the same case. It is because of this discrepancy that we present the chart of algorithm running times as a log-plot. Furthermore, due to the prohibitively high running time on the smaller data set, we omitted the divide-and-conquer algorithm for the experiment with the larger set.

The decline in accuracy for StreamKM++ at very low memory can be partially explained by the $\Theta(k^2 \log^8 n)$ points' worth of memory needed for a strong guarantee in previous theory work [12]. However, the fact that the algorithm is able to achieve a good approximation in practice while using far less than that amount of memory suggests that improved provable bounds for coreset algorithms may be on the horizon. We should note that the performance of the algorithm declines sharply as the memory difference with the authors' specification grows, but gains accuracy as the memory grows.

All three algorithms can be described as computing a weighted sketch of the data, and then solving $k$-means on that sketch. The final approximation ratios can be described as $\alpha(1 + \varepsilon)$ where $\alpha$ is the loss from the final batch algorithm. The coreset $\varepsilon$ is a direct function of the memory allowed to the algorithm, and can be made arbitrarily small. However, the memory needed to provably reduce $\varepsilon$ to a small constant is quite substantial, and while StreamKM++ does produce a good resulting clustering, it is not immediately clear the the discovery of better batch $k$-means algorithms would improve their solution quality. Our algorithm's $\varepsilon$ represents the ratio of the cost of our $\kappa$-mean solution to the cost of the optimum $k$-means solution. The provable value is a large constant, but since $\kappa$ is much larger than $k$, we would expect better performance in practice, and we observe this effect in our experiments.

For our algorithm, the observed value of $1 + \varepsilon$ has been typically between 1 and 3, whereas the D&C approach did not yield one better than 24, and was high (low thousands) for the very low memory conditions. The coreset algorithm was the worst, with even the best values in the mid ten figures (tens to hundreds of billions). The low ratio for our algorithm also suggests that our $\kappa$ facilities are a good sketch of the overall data, and thus our observed accuracy can be expected to improve as more accurate batch $k$-means algorithms are discovered.

### Acknowledgments

We are grateful to Christian Sohler's research group for providing their code for the `StreamKM++` algorithm. We also thank Jennifer Wortman Vaughan, Thomas G. Dietterich, Daniel Sheldon, Andrea Vattani, and Christian Sohler for helpful feedback on drafts of this paper. This work was done while all the authors were at UCLA; at that time, Adam Meyerson and Michael Shindler were partially supported by NSF CIF Grant CCF-1016540.

## Footnotes

[1] The BigCross dataset is 11,620,300 points in 57-dimensional space; it is available from [1]

[2] The Census1990 dataset is 2,458,285 points in 68 dimensions; it is available from [15]

[3]Visit http://web.engr.oregonstate.edu/~shindler/ to access code for our algorithms

## References

[1] http://www.cs.uni-paderborn.de/en/fachgebiete/ag-bloemer/research/clustering/streamkmpp.

[2] Marcel R. Ackermann, Christian Lammersen, Marcus Märtens, Christoph Raupach, Christian Sohler, and Kamil Swierkot. StreamKM++: A clustering algorithms for data streams. In *ALENEX*, 2010.

[3] Charu C. Aggarwal, editor. *Data Streams: Models and Algorithms.* Springer, 2007.

[4] Nir Ailon, Ragesh Jaiswal, and Claire Monteleoni. Streaming $k$-means approximation. In *NIPS*, 2009.

[5] Khaled Alsabti, Sanjay Ranka, and Vineet Singh. An efficient $k$-means clustering algorithm. In *HPDM*, 1998.

[6] Alexandr Andoni and Piotr Indyk. Near-optimal hashing algorithms for approximate nearest neighbor in high dimensions. *Communications of the ACM*, January 2008.

[7] David Arthur and Sergei Vassilvitskii. $k$-means++: The Advantages of Careful Seeding. In *SODA*, 2007.

[8] Vijay Arya, Naveen Garg, Rohit Khandekar, Adam Meyerson, Kamesh Munagala, and Vinayaka Pandit. Local search heuristic for $k$-median and facility location problems. In *STOC*, 2001.

[9] Vladimir Braverman, Adam Meyerson, Rafail Ostrovsky, Alan Roytman, Michael Shindler, and Brian Tagiku. Streaming $k$-means on Well-Clusterable Data. In *SODA*, 2011.

[10] Mihai Bădoiu, Sariel Har-Peled, and Piotr Indyk. Approximate clustering via core-sets. In *STOC*, 2002.

[11] Moses Charikar, Liadan O'Callaghan, and Rina Panigrahy. Better streaming algorithms for clustering problems. In *STOC*, 2003.

[12] Ke Chen. On coresets for $k$-median and $k$-means clustering in metric and euclidean spaces and their applications. *SIAM J. Comput.*, 2009.

[13] Dan Feldman, Morteza Monemizadeh, and Christian Sohler. A PTAS for $k$-means clustering based on weak coresets. In *SCG*, 2007.

[14] Gereon Frahling and Christian Sohler. Coresets in dynamic geometric data streams. In *STOC*, 2005.

[15] A. Frank and A. Asuncion. UCI machine learning repository, 2010.

[16] Sudipto Guha, Adam Meyerson, Nina Mishra, Rajeev Motwani, and Liadan O'Callaghan. Clustering data streams: Theory and practice. In *TDKE*, 2003.

[17] Sudipto Guha, Nina Mishra, Rajeev Motwani, and Liadan O'Callaghan. Clustering data streams. In *FOCS*, 2000.

[18] Sariel Har-Peled and Soham Mazumdar. On coresets for $k$-means and $k$-median clustering. In *STOC*, 2004.

[19] Anil Kumar Jain, M Narasimha Murty, and Patrick Joseph Flynn. Data clustering: a review. *ACM Computing Surveys*, 31(3), September 1999.

[20] Tapas Kanungo, David Mount, Nathan Netanyahu, Christine Piatko, Ruth Silverman, and Angela Wu. A local search approximation algorithm for $k$-means clustering. In *SCG*, 2002.

[21] Stuart Lloyd. Least Squares Quantization in PCM. In *Special issue on quantization, IEEE Transactions on Information Theory*, 1982.

[22] James MacQueen. Some methods for classification and analysis of multivariate observations. In *Berkeley Symposium on Mathematical Statistics and Probability*, 1967.

[23] Joel Max. Quantizing for minimum distortion. *IEEE Transactions on Information Theory*, 1960.

[24] Adam Meyerson. Online facility location. In *FOCS*, 2001.

[25] Rafail Ostrovsky, Yuval Rabani, Leonard Schulman, and Chaitanya Swamy. The Effectiveness of Lloyd-Type Methods for the $k$-Means Problem. In *FOCS*, 2006.

[26] Rina Panigrahy. Entropy based nearest neighbor search in high dimensions. In *SODA*, 2006.

[27] Dan Pelleg and Andrew Moore. Accelerating exact $k$-means algorithms with geometric reasoning. In *KDD*, 1999.

[28] Steven J. Phillips. Acceleration of $k$-means and related clustering problems. In *ALENEX*, 2002.

[29] Andrea Vattani. k-means requires exponentially many iterations even in the plane. *Discrete Computational Geometry*, June 2011.

